# Effective size of receptive fields of inferior temporal visual cortex neurons in natural scenes

**Thomas P. Trappenberg**
Dalhousie University
Faculty of Computer Science
5060 University Avenue, Halifax B3H 1W5, Canada
*tt@cs.dal.ca*

**Edmund T. Rolls and Simon M. Stringer**
University of Oxford,
Centre for Computational Neuroscience,
Department of Experimental Psychology,
South Parks Road, Oxford OX1 3UD, UK
*edmund.rolls,simon.stringer@psy.ox.ac.uk*

## Abstract

Inferior temporal cortex (IT) neurons have large receptive fields when a single effective object stimulus is shown against a blank background, but have much smaller receptive fields when the object is placed in a natural scene. Thus, translation invariant object recognition is reduced in natural scenes, and this may help object selection. We describe a model which accounts for this by competition within an attractor in which the neurons are tuned to different objects in the scene, and the fovea has a higher cortical magnification factor than the peripheral visual field. Furthermore, we show that top-down object bias can increase the receptive field size, facilitating object search in complex visual scenes, and providing a model of object-based attention. The model leads to the prediction that introduction of a second object into a scene with blank background will reduce the receptive field size to values that depend on the closeness of the second object to the target stimulus. We suggest that mechanisms of this type enable the output of IT to be primarily about one object, so that the areas that receive from IT can select the object as a potential target for action.

## 1 Introduction

Neurons in the macaque inferior temporal visual cortex (IT) that respond to objects or faces have large receptive fields when a single object or image is shown on an otherwise blank screen [1, 2, 3]. The responsiveness of the neurons to their effective stimuli independent of their position on the retina over many degrees is termed translation invariance. Translation invariant object recognition is an important property of visual processing, for it potentially enables the neurons that receive information from the inferior temporal visual cortex to

perform memory operations to determine whether for example the object has been seen before or is associated with reward independently of where the image was on the retina. This allows correct generalization over position, so that what is learned when an object is shown at one position on the retina generalizes correctly to other positions [4].

If more than one object is present on the screen, then there is evidence that the neuron responds more to the object at the fovea than in the parafoveal region [5, 6]. More recently, it has been shown that if an object is presented in a natural background (cluttered scene), and the monkey is searching for the object in order to touch it to obtain a reward, then the receptive fields are smaller than when the monkey performs the same task with the object against a blank background [7]. We define the size of a receptive field as twice the distance from the fovea (the centre of the receptive field) to locations at which the response decreases to half maximal. An analysis of IT neurons that responded to the target stimulus showed that the average size of the receptive fields shrinks from approximately 56 degrees in a blank background to approximately 12 degrees with a complex scene [8]. The responses of an IT cell with a large receptive field are illustrated in Figure 1A. There the average firing rates of the cell to an effective stimulus that the monkey had to touch on a touch-screen to receive reward is shown as a function of the angular distance of the object from the fovea. The solid line represents the results from experiments with the object placed in a blank background. This demonstrates the large receptive fields of IT cells that have often been reported in the literature [3]. In contrast, when the object is placed in a natural scene (cluttered background), the size of the receptive field is markedly smaller (dashed line).

## 2  The model

We formalized our understanding of how the dependence of the receptive field size on various conditions could be implemented in the ventral visual processing pathway by developing a neural network model with the components sufficient to produce the above effects. The model utilizes an attractor network representing the inferior temporal visual cortex, and a neural input layer with several retinotopically organized modules representing the visual scene in an earlier visual cortical area such as V4 (see Figure 1B). Each independent module within 'V4' represents a small part of the visual field and receives input from earlier visual areas represented by an input vector for each possible location which is unique for each object. Each module was 6 deg in width, matching the size of the objects presented to the network. For the simulations we chose binary random input vectors representing objects with $N^{\mathrm{V4}} a^{\mathrm{V4}}$ components set to ones and the remaining $N^{\mathrm{V4}}(1 - a^{\mathrm{V4}})$ components set to zeros. $N^{\mathrm{V4}}$ is the number of nodes in each module and $a^{\mathrm{V4}} = 0.2$ is the sparseness of the representation.

The structure labeled 'IT' represents areas of visual association cortex such as the inferior temporal visual cortex and cortex in the anterior part of the superior temporal sulcus in which neurons provide distributed representations of faces and objects [9, 3]. The activity $u_i^{\mathrm{IT}}(t)$ of nodes in this structure are governed by leaky integrator dynamics with time constant $\tau$

$$\tau \frac{\mathrm{d}u_i^{\mathrm{IT}}(t)}{\mathrm{d}t} = -u_i^{\mathrm{IT}}(t) + \sum_j (w_{ij}^{\mathrm{IT}} - c^{\mathrm{IT}}) r_j^{\mathrm{IT}}(t) + \sum_k w_{ik}^{\mathrm{IT-V4}} r_k^{\mathrm{V4}}(t) + k^{\mathrm{IT\_BIAS}} I_i^{\mathrm{OBJ}}. \quad (1)$$

The firing rate $r_i^{\mathrm{IT}}$ of the $i$th node is determined by a sigmoidal function from the activation $u_i^{\mathrm{IT}}$ as $r_i^{\mathrm{IT}}(t) = 1/(1 + \exp[-2\beta(u_i^{\mathrm{IT}}(t) - \alpha)])$, where the parameters $\beta = 1$ and $\alpha = 1$ represent the gain and the offset, respectively. The constant $c^{\mathrm{IT}}$ represents the strength of the activity-dependent global inhibition simulating the effects of inhibitory interneurons. The external 'top-down' input vector $I^{\mathrm{OBJ}}$ produces object-selective inputs, which are used as the attentional drive when a visual search task is simulated. The strength of this

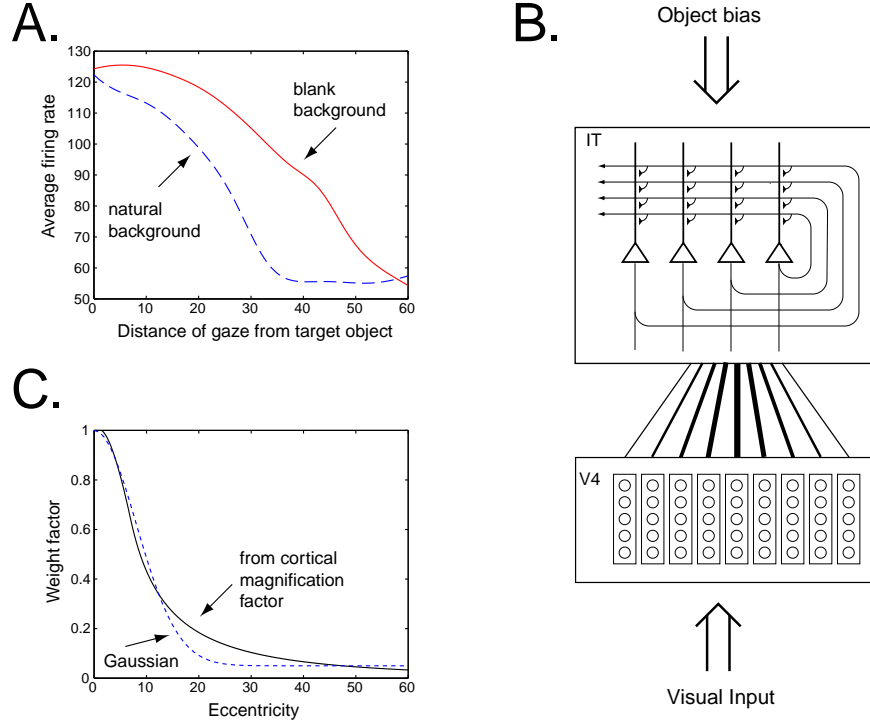

Figure 1: A) Average activity of a macaque inferior temporal cortex neuron as a function of the distance of the object from the fovea recorded in a visual search task when the object was in a blank or a cluttered natural background. B) Outline of the model used in this study with an attractor network labelled 'IT' that receives topgraphical organised inputs from an input neural layer labeled 'V4'. Objects close to the fovea produce stronger inputs to reflect the higher magnification factor of the visual representation close to the fovea. The attractor network also receives top-down object-based inputs, to incorporate object-based attention in a visual search task. C) The modulation factor used to weight inputs to IT from V4 shown as a function of their distance from the fovea. The values on the solid line are derived from cortical magnification factors, and were used in the simulations, whereas the dotted line corresponds to a shifted Gaussian function.

object bias is modulated by the value of $k^{\text{IT\_BIAS}}$. The recognition functionality of this structure is modeled as an attractor neural network (ANN) with trained memories indexed by $\mu$ representing particular objects. The memories are formed through Hebbian learning on sparse patterns,

$$w_{ij}^{\text{IT}} = k^{\text{IT}} \sum_{\mu} (\xi_i^{\mu} - a^{IT})(\xi_j^{\mu} - a^{IT}), \qquad (2)$$

where $k^{\text{IT}}$ (set to 1 in the simulations below) is a normalization constant that depends on the learning rate, $a^{IT} = 0.2$ is the sparseness of the training pattern in IT, and $\xi_i^{\mu}$ are the components of the pattern used to train the network. The weights $w_{ij}^{\text{IT}-\text{V4}}$ between the V4 nodes and IT nodes are trained by Hebbian learning of the form

$$w_{ij}^{\text{IT}-\text{V4}} = k^{\text{IT}-\text{V4}}(k) \sum_{\mu} (\xi_i^{\mu} - a^{V4})(\xi_j^{\mu} - a^{IT}). \qquad (3)$$

to produce object representations in IT based on inputs in V4. The normalizing modulation

factor $k^{\mathrm{IT-V4}}(k)$ allows the gain of inputs to be modulated as a function of their distance from the fovea, and depends on the module $k$ to which the presynaptic node belongs. The weight values between V4 and IT support translation invariant object recognition of a single object in the visual field if the normalization factor is the same for each module and the model is trained with the objects placed at every possible location in the visual field. The translation invariance of the weight vectors between each V4 module and the IT nodes is however explicitly modulated in our model by the module-dependent modulation factor $k^{\mathrm{IT-V4}}(k)$ as indicated in Figure 1B by the width of the lines connecting V4 with IT. The strength of the foveal module is strongest, and the strength decreases for modules representing increasing eccentricity. The form of this modulation factor was derived from the parameterization of the cortical magnification factors given by [10],[1] and is illustrated in Figure 1C as a solid line. Similar results to the ones presented here can be achieved with different forms of the modulation factor such as a shifted Gaussian as illustrated by the dashed line in Figure 1C.

## 3   Results

To study the ability of the model to recognize trained objects at various locations relative to the fovea we tested the network with distorted versions of the objects, and measured the 'correlation' between the target object and the final state of the attractor network. The correlation was estimated from the normalized dot product between the target object vector and the state of the IT network after a fixed amount of time sufficient for the network to settle into a stable state. The objects were always presented on backgrounds with some noise (introduced by flipping 2% of the bits in the scene) because the input to IT will inevitably be noisy under normal conditions of operation. All results shown in the following represent averages over 10 runs and over all patterns on which the network was trained.

### 3.1   Receptive fields are large in scenes with blank backgrounds

In the first experiments we placed only one object in the visual scene with different eccentricities relative to the fovea. The results of this simulation are shown in Figure 2A with the line labeled 'blank background'. The value of the object bias $k^{\mathrm{IT\_BIAS}}$ was set to 0 in these simulations. Good object retrieval (indicated by large correlations) was found even when the object was far from the fovea, indicating large IT receptive fields with a blank background. The reason that any drop is seen in performance as a function of eccentricity is because flipping 2% of the bits in the V4 modules introduces some noise into the recall process. The results demonstrate that the attractor dynamics can support translation invariant object recognition even though the weight vectors between V4 and IT are not translation invariant but are explicitly modulated by the modulation factor $k^{\mathrm{IT-V4}}$ derived from the cortical magnification factor.

### 3.2   The receptive field size is reduced in scenes with complex background

In a second experiment we placed individual objects at all possible locations in the visual scene representing natural (cluttered) visual scenes. The resulting correlations between the target pattern and asymptotic IT state are shown in Figure 2A with the line labeled 'natural background'. Many objects in the visual scene are now competing for recognition by the attractor network, while the objects around the foveal position are enhanced through the modulation factor derived by the cortical magnification factor. This results in a much smaller size of the receptive field of IT neurons when measured with objects in natural

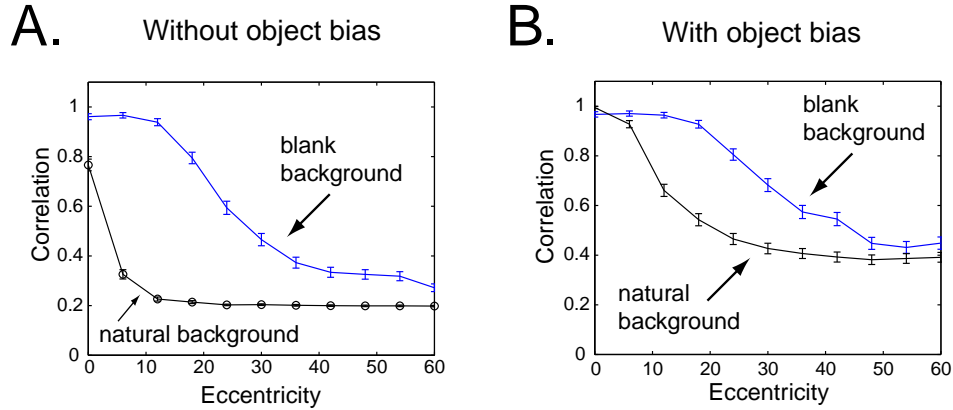

Figure 2: Correlations as measured by the normalized dot product between the object vector used to train IT and the state of the IT network after settling into a stable state with a single object in the visual scene (blank background) or with other trained objects at all possible locations in the visual scene (natural background). There is no object bias included in the results shown in graph A, whereas an object bias is included in the results shown in B with $k^{\mathrm{IT\_BIAS}} = 0.7$ in the experiments with a natural background and $k^{\mathrm{IT\_BIAS}} = 0.1$ in the experiments with a blank background.

backgrounds.

### 3.3 Object-based attention increases the receptive field size, facilitating object search in complex visual scenes

In addition to this major effect of the background on the size of the receptive field, which parallels and we suggest may account for the physiological findings outlined in the introduction, there is also a dependence of the size of the receptive fields on the level of object bias provided to the IT network. Examples are shown in Figure 2B where we used an object bias. The object bias biasses the IT network towards the expected object with a strength determined by the value of $k^{\mathrm{IT\_BIAS}}$, and has the effect of increasing the size of the receptive fields in both blank and natural backgrounds (see Figure 2B and compare to Figure 2A). This models the effect found neurophysiologically [8].[2]

### 3.4 A second object in a blank background reduces the receptive field size depending on the distance between the second object and the fovea

In the last set of experiments we placed two objects in an otherwise blank background. The IT network was biased towards one of the objects designated as the target object (in for example a visual search task), which was placed on one side of the fovea at different eccentricities from the fovea. The second object, a distractor object, was placed on the opposite side of the fovea at a fixed distance of $d$ degrees from the fovea. Results for different values of $d$ are shown in 3A. The results indicate that the size of the receptive field (for the target object) decreases with decreasing distance of the distractor object from the fovea. The size of the receptive fields (the width at half maximal response) is shown

in 3B. The size starts to increase linearly with increasing distance $d$ of the distractor object from the fovea until the influence of the distractor on the size of the receptive field levels off and approaches the value expected for the situation with one object in a visual scene and a blank background.

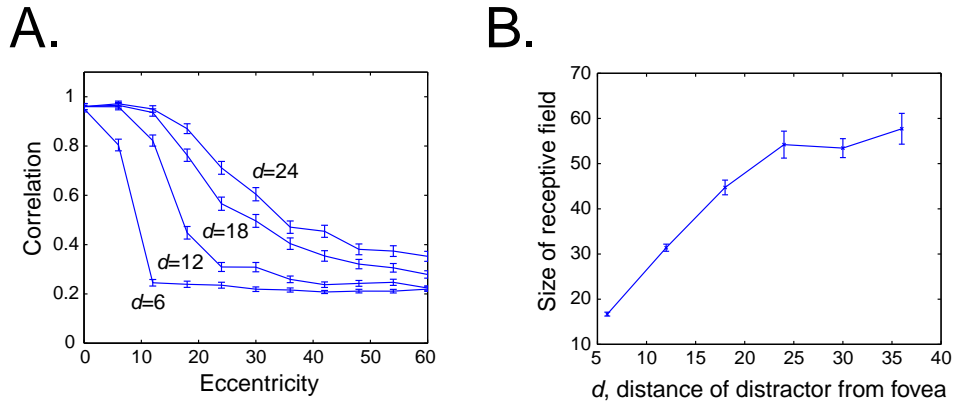

Figure 3: A) Correlations between the target object and the final state of the IT network in experiments with two objects in a visual scene with a blank background. The different curves correspond to different distances $d$ of the distractor object from the fovea. The eccentricity refers to the distance between the target object and the fovea. B) The size of the receptive field for the target as a function of the distance $d$ of the distractor object from the fovea.

## 4 Discussion

When single objects are shown in a scene with a blank background, the attractor network helps neurons to respond to an object with large eccentricities of this object relative to the fovea of the agent. When the object is presented in a natural scene, other neurons in the inferior temporal cortex become activated by the other effective stimuli present in the visual field, and these forward inputs decrease the response of the network to the target stimulus by a competitive process. The results found fit well with the neurophysiological data, in that IT operates with almost complete translation invariance when there is only one object in the scene, and reduces the receptive field size of its neurons when the object is presented in a cluttered environment.

The model here provides an explanation of the real IT neuronal responses in natural scenes and makes several predictions that can be explored experimentally. The model is compatible with the models developed by Gustavo Deco and colleagues (see, for example, [12, 13]) while specific simplifications and addition have been made to explore the variations in the size of receptive fields in IT.

The model accounts for the larger receptive field sizes from the fovea of IT neurons in natural backgrounds if the target is the object being selected compared to when it is not selected [8]. The model accounts for this by an effect of top-down bias which simply biasses the neurons towards particular objects compensating for their decreasing inputs produced by the decreasing magnification factor modulation with increasing distance from the fovea. Such object based attention signals could originate in the prefrontal cortex and could provide the object bias for the inferotemporal cortex [14].

We proposed that the effective variation of the size of the receptive field in the inferior

temporal visual cortex enables the brain areas that receive from this area (including the orbitofrontal cortex, amygdala, and hippocampal system) to read out the information correctly from the inferior temporal visual cortex about individual objects, because the neurons are responding effectively to the object close to the fovea, and respond very much less to objects away from the fovea.[3] This enables, for example, the correct reward association of an object to be determined by pattern association in the orbitofrontal cortex or amygdala, because they receive information essentially about the object at the fovea. Without this shrinkage in the receptive field size, the areas that receive from the inferior temporal visual cortex would respond to essentially all objects in a visual scene, and would therefore provide an undecipherable babel of information about all objects present in the visual scene. It appears that part of the solution to this potential binding problem that is used by the brain is to limit the size of the receptive fields of inferior temporal cortex neurons when natural environments are being viewed. The suggestion is that by providing an output about what is at the fovea in complex scenes, the inferior temporal visual cortex enables the correct reward association to be looked up in succeeding brain regions, and then for the object to be selected for action if appropriate. Part of the hypothesis here is that the coordinates of the object in the visual scene being selected for action are provided by the position in space to which the gaze is directed [7].

**Acknowledgments**

This research was supported by the Medical Research Council, grant PG9826105, and by the MRC Interdisciplinary Research Centre for Cognitive Neuroscience.

## Footnotes

[1]This parameterization is based on V1 data. However, it was shown that similar forms of the magnification factor hold also in V4 [11]

[2]The larger values of $k^{\mathrm{IT\_BIAS}}$ in the experiments with a natural background compared to the experiments in a blank background reflects the fact that more attention may be needed to find objects in natural cluttered environments.

[3]Note that it is possible that a "spotlight of attention" [15] can be moved away from the fovea, but at least during normal visual search tasks, the neurons are sensitive to the object at which the monkey is looking, that is which is on the fovea, as shown by [8]. Thus, spatial modulation of the responsiveness of neurons at the V4 level can be influenced by location-specific attentional modulations originating, for example, in the posterior parietal cortex, which may be involved in directing visual spatial attention [15].

[10] B.W. Dow, A.Z. Snyder, R.G. Vautin, and R. Bauer. Magnification factor and receptive field size in foveal striate cortex of the monkey. *Exp. Brain. Res.*, 44:213:228, 1981.

[11] R. Gattass, A.P.B. Sousa, and E. Covey. Cortical visual areas of the macaque: Possible substrates for pattern recognition mechanisms. *Exp. Brain. Res.*, Supplement 11, 1985.

[12] G. Deco and J. Zihl. Top-down selective visual attention: A neurodynamical approach. *Visual Cognition*, 8:119–140, 2001.

[13] E. T. Rolls and G. Deco. Computational neuroscience of vision. Oxford University Press, Oxford, 2002.

[14] A. Renart, N. Parga, and E. T. Rolls. A recurrent model of the interaction between the prefrontal cortex and inferior temporal cortex in delay memory tasks. In S.A. Solla, T.K. Leen, and K.-R. Mueller, editors, *Advances in Neural Information Processing Systems*. MIT Press, Cambridge Mass, 2000. in press.

[15] R. Desimone and J. Duncan. Neural mechanisms of selective visual attention. *Annual Review of Neuroscience*, 18:193–222, 1995.

# References

[1] C. G. Gross, R. Desimone, T. D. Albright, and Schwartz E. L. Inferior temporal cortex and pattern recognition. *Experimental Brain Research*, 11:179–201, 1985.

[2] M. J. Tovee, E. T. Rolls, and P. Azzopardi. Translation invariance and the responses of neurons in the temporal visual cortical areas of primates. *Journal of Neurophysiology*, 72:1049–1060, 1994.

[3] E. T. Rolls. Functions of the primate temporal lobe cortical visual areas in invariant visual object and face recognition. *Neuron*, 27:205– 218, 2000.

[4] E. T. Rolls and A. Treves. *Neural Networks and Brain Function*. Oxford University Press, Oxford, 1998.

[5] T. Sato. Interactions of visual stimuli in the receptive fields of inferior temporal neurons in macaque. *Experimental Brain Research*, 77:23–30, 1989.

[6] E. T. Rolls and M. J. Tovee. The responses of single neurons in the temporal visual cortical areas of the macaque when more than one stimulus is present in the visual field. *Experimental Brain Research*, 103:409–420, 1995.

[7] E. T. Rolls, B. Webb, and M. C. A. Booth. Responses of inferior temporal cortex neurons to objects in natural scenes. *Society for Neuroscience Abstracts*, 26:1331, 2000.

[8] E. T. Rolls, F. Zheng, and N. Aggelopoulos. Responses of inferior temporal cortex neurons to objects in natural scenes. *Society for Neuroscience Abstracts*, 27, 2001.

[9] M. C. A. Booth and E. T. Rolls. View-invariant representations of familiar objects by neurons in the inferior temporal visual cortex. *Cerebral Cortex*, 8:510–523, 1998.

